# Mass meta-analysis in Talairach space

**Finn Årup Nielsen**
Neurobiology Research Unit, Rigshospitalet
Copenhagen, Denmark
and
Informatics and Mathematical Modelling, Technical University of Denmark,
Lyngby, Denmark
fn@imm.dtu.dk

## Abstract

We provide a method for mass meta-analysis in a neuroinformatics database containing stereotaxic Talairach coordinates from neuroimaging experiments. Database labels are used to group the individual experiments, e.g., according to cognitive function, and the consistent pattern of the experiments within the groups are determined. The method voxelizes each group of experiments via a kernel density estimation, forming probability density volumes. The values in the probability density volumes are compared to null-hypothesis distributions generated by resamplings from the entire unlabeled set of experiments, and the distances to the null-hypotheses are used to sort the voxels across groups of experiments. This allows for mass meta-analysis, with the construction of a list with the most prominent associations between brain areas and group labels. Furthermore, the method can be used for functional labeling of voxels.

## 1   Introduction

Neuroimaging experimenters usually report their results in the form of 3-dimensional coordinates in the standardized stereotaxic Talairach system [1]. Automated meta-analytic and information retrieval methods are enabled when such data are represented in databases such as the BrainMap DBJ ([2], www.brainmapdbj.org) or the Brede database [3]. Example methods include outlier detection [4] and identification of similar volumes [5].

Apart from the stereotaxic coordinates, the databases record details of the experimental situation, e.g., the behavioral domain and the scanning modality. In the Brede database the main annotation is the so-called "external components"[1] which are heuristically organized in a simple ontology: A directed graph (more specifically, a causal network) with the most general components as the roots of the graph, e.g.,

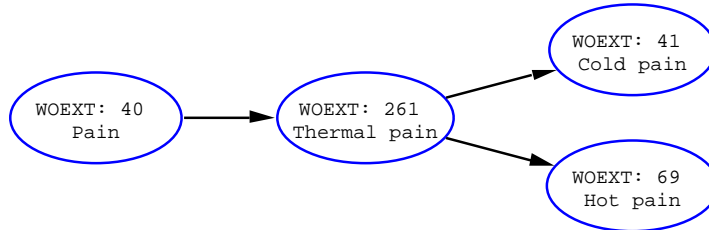

Figure 1: The external components around "thermal pain" with "pain" as the parent of "thermal pain" and "cold pain" and "hot pain" as children.

"hot pain" is a child of "thermal pain" that in turn is a child of "pain" (see Figure 1). The simple ontology is setup from information typically found in the introduction section of scientific articles, and it is compared with the Medical Subject Headings ontology of the National Library of Medicine. The ontology is stored in a simple XML file.

The Brede database is organized, like the BrainMap DBJ, on different levels with scientific papers on the highest level. Each scientific paper contains one or more "experiments", which each in turn contains one or more locations. The individual experiments are typically labeled with an external component. The experiments that are labeled with the same external component form a group, and the distribution of locations within the group become relevant: If a specific external component is localized to a specific brain region, then the locations associated with the external component should cluster in Talairach space.

We will describe a meta-analytic method that identifies important associations between external components and clustered Talairach coordinates. We have previously modeled the relation between Talairach coordinates and neuroanatomical terms [4, 6] and the method that we propose here can be seen as an extension describing the relationship between Talairach coordinates and, e.g., cognitive components.

## 2  Method

The data from the Brede database [3] was used, which at the time contained data from 126 scientific article containing 391 experiments and 2734 locations. There were 380 external components. The locations referenced with respect to the MNI atlas were realigned to the Talairach atlas [7].

To form a vectorial representation, each location was voxelized by convolving the location $l$ at position $\mathbf{v}_l = [x, y, z]'$ with a Gaussian kernel [4, 8, 9]. This constructed a probability density in Talairach space $\mathbf{v}$

$$p(\mathbf{v}|l) = (2\pi\sigma^2)^{-3/2} \exp\left[-\frac{(\mathbf{v} - \mathbf{v}_l)'(\mathbf{v} - \mathbf{v}_l)}{2\sigma^2}\right], \qquad (1)$$

with the width $\sigma$ fixed to 1 centimeter. To form a resulting probability density volume $p(\mathbf{v}|t)$ for an external component $t$ the individual components from each location were multiplied by the appropriate priors and summed

$$p(\mathbf{v}|t) = \sum_{l,e} p(\mathbf{v}|l) \, P(l|e) \, P(e|t), \qquad (2)$$

with $P(l|e) = 0$ if the $l$ location did not appear in the $e$ experiment and $P(e|t) = 0$ if the $e$ experiment is not associated with the $t$ external components. The precise

normalization of these priors is an unresolved problem. A paper with many locations and experiments should not be allowed to dominate the results. This can be the case if all locations are given equal weight. On the other hand a paper which reports just a single coordinate should probably not be weighted as much as one with many experiments and locations: Few reported locations might be due to limited (statistical) power of the experiment. As a compromise between the two extremes we used the square root of the number of the locations within an experiment and the square root of the number of experiments within a paper for the prior $P(l|e)$. The square root normalization is also an appropriate normalization in certain voting systems [10]. The second prior was uniform $P(e|t) \propto 1$ for those experiments that were labeled with the $t$ external component.

The continuous volume were sampled at regular grid points to establish a vector $\mathbf{w}_t$ for each external component

$$\mathbf{w}_t \equiv p(\mathbf{v}|t). \tag{3}$$

Null-hypothesis distributions for the maximum statistics $u$ across the voxels in the volume were built up by resampling: A number of experiments $E$ was selected and $E$ experiments were resampled, with replacement, from the entire set of 391 experiments, ignoring the grouping imposed by the external component labeling. The experiments were resampled without regard to the paper they originated from. The maximum across voxels was found as:

$$u_r(E) = \max_j \left[ w_r(j) \right], \tag{4}$$

where $j$ is an index over voxels and $r$ is the resample index. With $R$ resamplings we obtain a vector $\mathbf{u}(E) = [u_1(E) \ldots u_r(E) \ldots u_R(E)]$ that is a function of the number of experiments and which forms an empirical distribution $u(E)$. When the value $w_{t,j}$ of the $j$ voxel of the $t$ external component was compared with the distribution, a distance to the null-hypothesis can be generated

$$d_{t,j} = \text{Prob} \left[ w_{t,j} > u(E_t) \right], \tag{5}$$

where $1 - d$ is a statistical $P$-value and where $E_t$ is the number of experiment associated with the $t$ external component. Thus the resampling allows us to convert the probability density value to a probability that is comparable across external components of different sizes. The maximum statistics deal automatically with the multiple comparison problem across voxels [11].

$d_{t,j}$ can be computed by counting the fraction of the resampled values $u_r$ that are below the value of $w_{t,j}$. The resampling distribution can also be approximated and smoothed by modeling it with a non-linear function. In our case we used a standard two-layer feed-forward neural network with hyperbolic tangent hidden units [12, 13] modeling the function $f(E, u) = \text{atanh}(2d - 1)$ with a quadratic cost function. The non-linear function allows for a more compact representation of the empirical distribution of the resampled maximum statistics.

As a final step, the probability volumes for the external components $\mathbf{w}_t$ were thresholded on selected levels and isosurfaces generated in the distance volume for visualization. Connected voxels within the thresholded volume were found by region identification and the local maxima in the regions were determined.

Functional labeling of specified voxels is also possible: The distances $d_{t,j}$ were collected in a (external component $\times$ voxel)-matrix $\mathbf{D}$ and the elements in the $j$ column sorted. Lastly, the voxel were labeled with the top external component.

Only the bottom nodes of the causal networks of external components are likely to be directly associated with experiments. To label the ancestors, the labels from

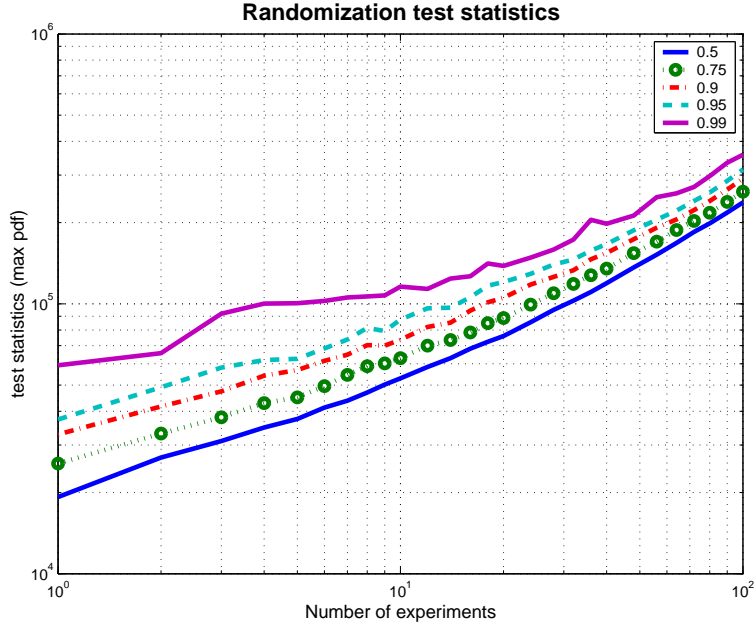

Figure 2: The test statistics at various distances to the null-hypothesis $(d = 1 - P)$ after 1000 resamplings. The distance is shown as a function of the number of experiments $E$ in the resampling.

their descendants were back propagated, e.g., a study explicitly labeled as "hot pain" were also be labeled as "thermal pain" and "pain". Apart from this simple back propagation the hierarchical structure of the external components was not incorporated into the prior.

## 3   Results

Figure 2 shows isolines in the cumulative distribution of the resampled maximum statistics $u(E)$ as a function of the resampling set size (number of experiments) from $E = 1$ to $E = 100$. Since the vectorized volume is not normalized to form a probability density the curves are increasing with our selected normalization.

Table 1 shows the result of sorting the maximum distances across voxel within the external components. Topping the list are external components associated with movement. The voxel with the largest distance is localized in $\mathbf{v} = (0, -8, 56)$ which most likely is due to movement studies activating the supplementary motor area. In the Brede database the mean is $(6, -7, 55)$ for the locations in the right hemisphere labeled as supplementary motor area. Other voxels with a high distance for the movement external components are located in the primary motor area.

A number of other entries on the list are associated with pain, with the main voxel at $(0, 8, 32)$ in the right anterior cingulate. Other important areas are shown in Figure 3 with isosurfaces in the distance volume for the external component "pain" (WOEXT: 40). These are localized in the anterior cingulate, right and left insula and thalamus.

Other external components high on the list are "audition" together with "voice"

| # | $d$ | $x$ | $y$ | $z$ | Name (WOEXT) |
|---|-----|-----|-----|-----|--------------|
| 1 | 1.00 | 0 | −8 | 56 | Localized movement (266) |
| 2 | 1.00 | 0 | −8 | 56 | Motion, movement, locomotion (4) |
| 3 | 1.00 | 0 | 8 | 32 | Pain (40) |
| 4 | 1.00 | 0 | 8 | 32 | Thermal pain (261) |
| 5 | 1.00 | 56 | −16 | 0 | Audition (14) |
| 6 | 1.00 | 0 | 8 | 32 | Temperature sensation (204) |
| 7 | 1.00 | 0 | 8 | 32 | Somesthesis (17) |
| 8 | 0.99 | 0 | −56 | 16 | Memory retrieval (24) |
| 9 | 0.99 | 0 | 8 | 32 | Warm temperature sensation (207) |
| 10 | 0.99 | 24 | −8 | −8 | Unpleasantness (153) |
| 11 | 0.99 | 56 | −16 | 0 | Voice (167) |
| 12 | 0.99 | 0 | −56 | 16 | Memory (9) |
| 13 | 0.99 | 24 | −8 | −8 | Emotion (3) |
| 14 | 0.99 | 0 | −56 | 16 | Long-term memory (112) |
| 15 | 0.99 | 0 | −56 | 16 | Declarative memory (319) |

Table 1: The top 15 elements of the list, showing the external components that score the highest, the distance to the null-hypothesis $d$, and the associated Talairach $x$, $y$ and $z$ coordinates. The numbers in the parentheses are the Brede database identifiers for the external components (WOEXT). This list was generated with coarse $8 \times 8 \times 8 \text{mm}^3$ voxels and using the non-linear model approximation for the cumulative distribution functions.

appearing in left and right superior temporal gyrus, and memory emerging in the posterior cingulate area. Unpleasantness and emotion are high on the list due to, e.g., "fear" and "disgust" experiments that report activation in the right amygdala and nearby areas.

An example of the functional labeling of a voxel appears in Table 2. The chosen voxel is $(0, −56, 16)$ that appears in the posterior cingulate. Memory retrieval is the first on the list in accordance with Table 1. Many of the other external components on the list are also related to memory.

## 4   Discussion

The Brede database contains many thermal pain experiments, and it causes high scores for voxels from external components such as "pain" and "thermal pain". The four focal "brain activations" that appear in Figure 3 are localized in areas (anterior cingulate, insula and thalamus) that an expert reviewer has previously identified as important in pain [14]. Thus there is consistency between our automated meta-analytic technique and a "manual" expert review.

Many experiments that report activation in the posterior cingulate area have been included in the Brede database, and this is probably why memory is especially associated with this area. A major review of 275 functional neuroimaging studies found that episodic memory retrieval is the cognitive function with highest association with the posterior cingulate [15], so our finding is again in alignment with an

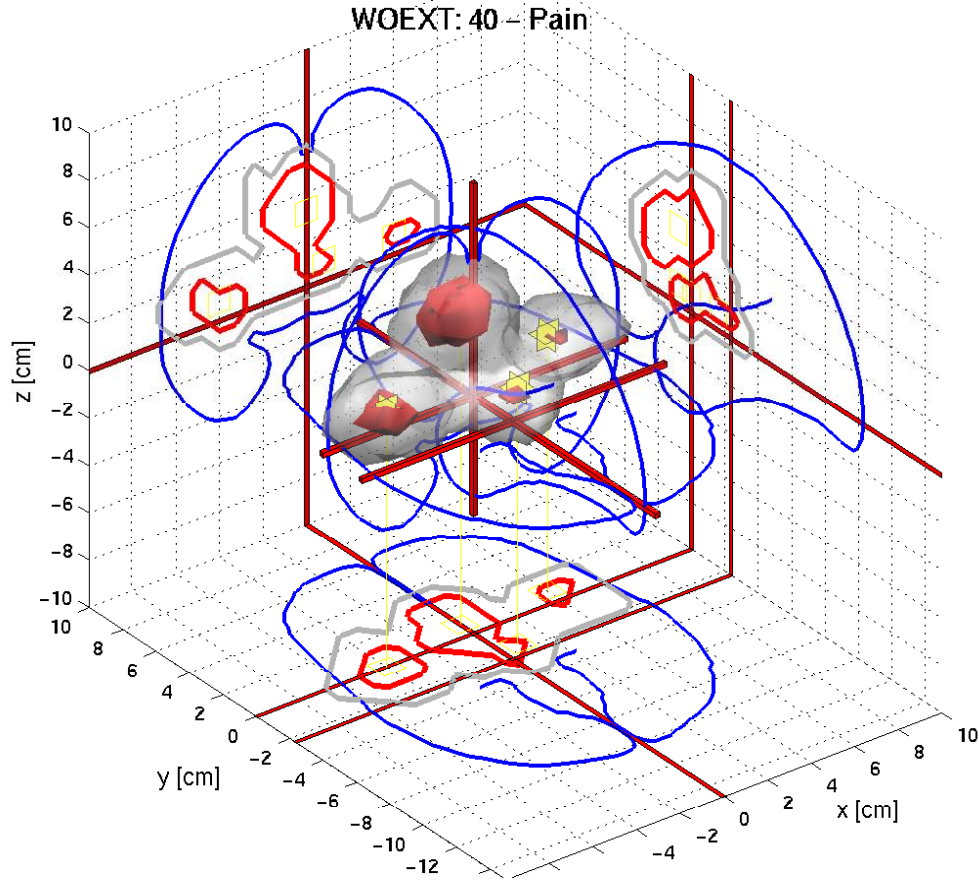

Figure 3: Plot of the important areas associated with the external component "pain". The red opaque isosurface is on the level $d = 0.95$ in the distance volume while the gray transparent surface appears at $d = 0.05$. Yellow glyphs appear at the local maxima in the thresholded volume. The viewpoint is situated nearest to the left superior posterior corner of the brain.

expert review.

A number of the substantial associations between brain areas and external components are not surprising, e.g., audition associating with superior temporal gyrus. Our method has no inherent knowledge of what is already known, and thus not able distinguish novel associations from trivial.

A down-side with the present method is that it requires the labeling of experiments during database entry and the construction of the hierarchy of the labels (Figure 1). Both are prone to "interpretation" and this is particularly a problem for complex cognitive functions. Our methodology, however, does not necessarily impose a single organization of the external components, and it is possible to rearrange these by defining another adjacency matrix for the graph.

In Table 1 the brain areas are represented in terms of Talairach coordinates. It should be possible to convert these coordinates further to neuroanatomical terms

| # | $d$ | Name (WOEXT) |
|---|------|---------------|
| 1 | 0.99 | Memory retrieval (24) |
| 2 | 0.99 | Memory (9) |
| 3 | 0.99 | Long-term memory (112) |
| 4 | 0.99 | Declarative memory (319) |
| 5 | 0.99 | Episodic memory (49) |
| 6 | 0.96 | Autobiographical memory (259) |
| 7 | 0.94 | Cognition (2) |
| 8 | 0.94 | Episodic memory retrieval (109) |
| 9 | 0.58 | Disease (79) |
| 10 | 0.16 | Recognition (190) |
| 11 | 0.14 | Psychiatric disorders (82) |
| 12 | 0.14 | Neurotic, stress and somatoform disorders (227) |
| 13 | 0.11 | Severe stress reactions and adjustment disorders (228) |
| 14 | 0.09 | Emotion (3) |
| 15 | 0.02 | Semantic memory (318) |

Table 2: Example of a functional label list of a voxel $\mathbf{v} = (0, -56, 16)$ in the posterior cingulate area.

by using the models between coordinates and lobar anatomy that we previously have established [4, 6].

Functional labeling should allow us to build a complete functional atlas for the entire brain. The utility of this approach is, however, limited by the small size of the Brede database and its bias towards specific brain regions and external components. But such a functional atlas will serve as a neuroinformatic organizer for the increasing number of neuroimaging studies.

**Acknowledgment**

I am grateful to Matthew G. Liptrot for reading and commenting on the manuscript. Lars Kai Hansen is thanked for discussion, Andrew C. N. Chen for identifying some of the thermal pain studies and the Villum Kann Rasmussen Foundation for their generous support of the author.

## Footnotes

[1] External components might be thought of as "cognitive components" or simply "brain functions", but they are more general, e.g., they also incorporate neuroreceptors. The components are called "external" since they are external variables to the brain image.

# References

[1] Jean Talairach and Pierre Tournoux. *Co-planar Stereotaxic Atlas of the Human Brain*. Thieme Medical Publisher Inc, New York, January 1988.

[2] Peter T. Fox and Jack L. Lancaster. Mapping context and content: the Brain-Map model. *Nature Reviews Neuroscience*, 3(4):319–321, April 2002.

[3] Finn Årup Nielsen. The Brede database: a small database for functional neuroimaging. *NeuroImage*, 19(2), June 2003. Presented at the 9th International Conference on Functional Mapping of the Human Brain, June 19–22, 2003, New York, NY. Available on CD-Rom.

[4] Finn Årup Nielsen and Lars Kai Hansen. Modeling of activation data in the BrainMap$^{TM}$ database: Detection of outliers. *Human Brain Mapping*, 15(3):146–156, March 2002.

[5] Finn Årup Nielsen and Lars Kai Hansen. Finding related functional neuroimaging volumes. *Artificial Intelligence in Medicine*, 30(2):141–151, February 2004.

[6] Finn Årup Nielsen and Lars Kai Hansen. Automatic anatomical labeling of Talairach coordinates and generation of volumes of interest via the Brain-Map database. *NeuroImage*, 16(2), June 2002. Presented at the 8th International Conference on Functional Mapping of the Human Brain, June 2–6, 2002, Sendai, Japan. Available on CD-Rom.

[7] Matthew Brett. The MNI brain and the Talairach atlas. http://www.mrc-cbu.cam.ac.uk/Imaging/mnispace.html, August 1999. Accessed 2003 March 17.

[8] Peter E. Turkeltaub, Guinevere F. Eden, Karen M. Jones, and Thomas A. Zeffiro. Meta-analysis of the functional neuroanatomy of single-word reading: method and validation. *NeuroImage*, 16(3 part 1):765–780, July 2002.

[9] J. M. Chein, K. Fissell, S. Jacobs, and Julie A. Fiez. Functional heterogeneity within Broca's area during verbal working memory. *Physiology & Behavior*, 77(4-5):635–639, December 2002.

[10] Lionel S. Penrose. The elementary statistics of majority voting. *Journal of the Royal Statistical Society*, 109:53–57, 1946.

[11] Andrew. P. Holmes, R. C. Blair, J. D. G. Watson, and I. Ford. Non-parametric analysis of statistic images from functional mapping experiments. *Journal of Cerebral Blood Flow and Metabolism*, 16(1):7–22, January 1996.

[12] Claus Svarer, Lars Kai Hansen, and Jan Larsen. On the design and evaluation of tapped-delay lines neural networks. In *Proceedings of the IEEE International Conference on Neural Networks, San Francisco, California, USA*, volume 1, pages 46–51, 1993.

[13] Lars Kai Hansen, Finn Årup Nielsen, Peter Toft, Matthew George Liptrot, Cyril Goutte, Stephen C. Strother, Nicholas Lange, Anders Gade, David A. Rottenberg, and Olaf B. Paulson. "lyngby" — a modeler's Matlab toolbox for spatio-temporal analysis of functional neuroimages. *NeuroImage*, 9(6):S241, June 1999.

[14] Martin Ingvar. Pain and functional imaging. *Philosophical Transactions of the Royal Society of London. Series B, Biological Sciences*, 354(1387):1347–1358, July 1999.

[15] Roberto Cabeza and Lars Nyberg. Imaging cognition II: An empirical review of 275 PET and fMRI studies. *Journal of Cognitive Neuroscience*, 12(1):1–47, January 2000.
